# Spatial Normalized Gamma Processes

**Vinayak Rao**
Gatsby Computational Neuroscience Unit
University College London
vrao@gatsby.ucl.ac.uk

**Yee Whye Teh**
Gatsby Computational Neuroscience Unit
University College London
ywteh@gatsby.ucl.ac.uk

## Abstract

Dependent Dirichlet processes (DPs) are dependent sets of random measures, each being marginally DP distributed. They are used in Bayesian nonparametric models when the usual exchangeability assumption does not hold. We propose a simple and general framework to construct dependent DPs by marginalizing and normalizing a single gamma process over an extended space. The result is a set of DPs, each associated with a point in a space such that neighbouring DPs are more dependent. We describe Markov chain Monte Carlo inference involving Gibbs sampling and three different Metropolis-Hastings proposals to speed up convergence. We report an empirical study of convergence on a synthetic dataset and demonstrate an application of the model to topic modeling through time.

## 1   Introduction

Bayesian nonparametrics have recently garnered much attention in the machine learning and statistics communities, due to their elegant treatment of infinite dimensional objects like functions and densities, as well as their ability to sidestep the need for model selection. The Dirichlet process (DP) [1] is a cornerstone of Bayesian nonparametrics, and forms a basic building block for a wide variety of extensions and generalizations, including the infinite hidden Markov model [2], the hierarchical DP [3], the infinite relational model [4], adaptor grammars [5], to name just a few.

By itself, the DP is a model that assumes that data are infinitely exchangeable, i.e. the ordering of data items does not matter. This assumption is false in many situations and there has been a concerted effort to extend the DP to more structured data. Much of this effort has focussed on defining priors on collections of dependent random probability measures. [6] expounded on the notion of *dependent DPs*, that is, a dependent set of random measures that are all marginally DPs. The property of being marginally DP here is both due to a desire to construct mathematically elegant solutions, and also due to the fact that the DP and its implications as a statistical model, e.g. on the behaviour of induced clusterings of data or asymptotic consistency, are well-understood. In this paper, we propose a simple and general framework for the construction of dependent DPs on arbitrary spaces. The idea is based on the fact that just as Dirichlet distributions can be generated by drawing a set of independent gamma variables and normalizing, the DP can be constructed by drawing a sample from a *gamma process* ($\Gamma$P) and normalizing (i.e. it is an example of a normalized random measure [7, 8]). A $\Gamma$P is an example of a *completely random measure* [9]: it has the property that the random masses it assigns to disjoint subsets are independent. Furthermore, the restriction of a $\Gamma$P to a subset is itself a $\Gamma$P. This implies the following easy construction of a set of dependent DPs: define a $\Gamma$P over an extended space, associate each DP with a different region of the space, and define each DP by normalizing the restriction of the $\Gamma$P on the associated region. This produces a set of dependent DPs, with the amount of overlap among the regions controlling the amount of dependence. We call this model a *spatial normalized gamma process* (SN$\Gamma$P). More generally, our construction can be extended to normalizing restrictions of any completely random measure, and we call the resulting dependent random measures *spatial normalized random measures* (SNRMs).

In Section 2 we briefly describe the ΓP. Then we describe our construction of the SNΓP in Section 3. We describe inference procedures based on Gibbs and Metropolis-Hastings sampling in Section 4 and report experimental results in Section 5. We conclude by discussing limitations and possible extensions of the model as well as related work in Section 6.

## 2 Gamma Processes

We briefly describe the gamma process (ΓP) here. A good high-level introduction can be found in [10]. Let $(\Theta, \Omega)$ be a measure space on which we would like to define a ΓP. Like the DP, realizations of the ΓP are atomic measures with random weighted point masses. We can visualize the point masses $\theta \in \Theta$ and their corresponding weights $w > 0$ as points in a product space $\Theta \otimes [0, \infty)$. Consider a Poisson process over this product space with mean measure

$$\mu(d\theta dw) = \alpha(d\theta) w^{-1} e^{-w} dw. \tag{1}$$

Here $\alpha$ is a measure on the space $(\Theta, \Omega)$ and is called the *base measure* of the ΓP. A sample from this Poisson process will yield an infinite set of atoms $\{\theta_i, w_i\}_{i=1}^{\infty}$ since $\int_{\Theta \otimes [0,\infty)} \mu(d\theta dw) = \infty$. A sample from the ΓP is then defined as

$$G = \sum_{i=1}^{\infty} w_i \delta_{\theta_i} \sim \Gamma P(\alpha). \tag{2}$$

It can be shown that the total mass $G(S) = \sum_{i=1}^{\infty} w_i \mathbf{1}(\theta_i \in S)$ of any measurable subset $S \subset \Theta$ is simply gamma distributed with shape parameter $\alpha(S)$, thus the natural name *gamma* process. Dividing $G$ by $G(\Theta)$, we get a *normalized random measure*—a random probability measure. Specifically, we get a sample from the Dirichlet process $\text{DP}(\alpha)$:

$$D = G/G(\Theta) \sim \text{DP}(\alpha). \tag{3}$$

Here we used an atypical parameterization of the DP in terms of the base measure $\alpha$. The usual (equivalent) parameters of the DP are: strength parameter $\alpha(\Theta)$ and base distribution $\alpha/\alpha(\Theta)$. Further, the DP is independent of the normalization: $D \perp\!\!\!\perp G(\Theta)$.

The gamma process is an example of a *completely random measure* [9]. This means that for mutually disjoint measurable subsets $S_1, \ldots, S_n \in \Omega$ the random numbers $\{G(S_1), \ldots, G(S_n)\}$ are mutually independent. Two straightforward consequences will be of importance in the rest of this paper. Firstly, if $S \in \Omega$ then the restriction $G'(d\theta) = G(d\theta \cap S)$ onto $S$ is a ΓP with base measure $\alpha'(d\theta) = \alpha(d\theta \cap S)$. Secondly, if $\Theta = \Theta_1 \otimes \Theta_2$ is a two dimensional space, then the projection $G''(d\theta_1) = \int_{\Theta_2} G(d\theta_1 d\theta_2)$ onto $\Theta_1$ is also a ΓP with base measure $\alpha''(d\theta_1) = \int_{\Theta_2} \alpha(d\theta_1 d\theta_2)$.

## 3 Spatial Normalized Gamma Processes

In this section we describe our proposal for constructing dependent DPs. Let $(\Theta, \Omega)$ be a probability space and $\mathbb{T}$ an index space. We wish to construct a set of dependent random measures over $(\Theta, \Omega)$, one $D_t$ for each $t \in \mathbb{T}$ such that each $D_t$ is marginally DP. Our approach is to define a gamma process $G$ over an extended space and let each $D_t$ be a normalized restriction/projection of $G$. Because restrictions and projections of gamma processes are also gamma processes, each $D_t$ will be DP distributed.

To this end, let $\mathbb{Y}$ be an auxiliary space and for each $t \in \mathbb{T}$, let $Y_t \subset \mathbb{Y}$ be a measurable set. For any measure $\mu$ over $\Theta \otimes \mathbb{Y}$ define the *restricted projection* $\mu_t$ by

$$\mu_t(d\theta) = \int_{Y_t} \mu(d\theta dy) = \mu(d\theta \otimes Y_t). \tag{4}$$

Note that $\mu_t$ is a measure over $\Theta$ for each $t \in \mathbb{T}$. Now let $\alpha$ be a base measure over the product space $\Theta \otimes \mathbb{Y}$ and consider a gamma process

$$G \sim \Gamma P(\alpha) \tag{5}$$

over $\Theta \otimes \mathbb{Y}$. Since restrictions and projections of $\Gamma$Ps are $\Gamma$Ps as well, $G_t$ will be a $\Gamma$P over $\Theta$ with base measure $\alpha_t$:

$$G_t(d\theta) = \int_{Y_t} G(d\theta dy) \sim \Gamma P(\alpha_t) \tag{6}$$

Now normalizing,

$$D_t = G_t/G_t(\Theta) \sim DP(\alpha_t) \tag{7}$$

We call the resulting set of dependent DPs $\{D_t\}_{t \in \mathbb{T}}$ *spatial normalized gamma processes* (SN$\Gamma$Ps). If the index space is continuous, $\{D_t\}_{t \in \mathbb{T}}$ can equivalently be thought of as a *measure-valued stochastic process*. The amount of dependence between $D_s$ and $D_t$ for $s, t \in \mathbb{T}$ is related to the amount of overlap between $Y_s$ and $Y_t$. Generally, the subsets $Y_t$ are defined so that the closer $s$ and $t$ are in $\mathbb{T}$, the more overlap $Y_s$ and $Y_t$ have and as a result $D_s$ and $D_t$ are more dependent.

## 3.1 Examples

We give two examples of SN$\Gamma$Ps, both with index set $\mathbb{T} = \mathbb{R}$ interpreted as the time line. Generalizations to higher dimensional Euclidean spaces $\mathbb{R}^n$ are straightforward. Let $H$ be a base distribution over $\Theta$ and $\gamma > 0$ be a concentration parameter.

The first example uses $\mathbb{Y} = \mathbb{R}$ as well, with the subsets being $Y_t = [t - L, t + L]$ for some fixed window length $L > 0$. The base measure is $\alpha(d\theta dy) = \gamma H(d\theta) dy/2L$. In this case the measure-valued stochastic process $\{D_t\}_{t \in \mathbb{R}}$ is stationary. The base measure $\alpha_t$ works out to be:

$$\alpha_t(d\theta) = \int_{t-L}^{t+L} \gamma H(d\theta) dy/2L = \gamma H(d\theta), \tag{8}$$

so that each $D_t \sim DP(\gamma H)$ with concentration parameter $\gamma$ and base distribution $H$. We can interpret this SN$\Gamma$P as follows. Each atom in the overall $\Gamma$P $G$ has a time-stamp $y$ and a time-span of $[y - L, y + L]$, so that it will only appear in the DPs $D_t$ within the window $t \in [y - L, y + L]$. As a result, two DPs $D_s$ and $D_t$ will share more atoms the closer $s$ and $t$ are to each other, and no atoms if $|s - t| > 2L$. Further, the dependence between $D_s$ and $D_t$ depends on $|s - t|$ only, decreasing with increasing $|s - t|$ and independent if $|s - t| > 2L$.

The second example generalizes the first one by allowing different atoms to have different window lengths. Each atom now has a time-stamp $y$ and a window length $l$, so that it appears in DPs in the window $[y - l, y + l]$. Our auxiliary space is thus $\mathbb{Y} = \mathbb{R} \otimes [0, \infty)$, with $Y_t = \{(y, l) : |y - t| \leq l\}$ (see Figure 1). Let $\beta(dl)$ be a distribution over window lengths in $[0, \infty)$. We use the base measure $\alpha(d\theta dy dl) = \gamma H(d\theta) dy \beta(dl)/2l$. The restricted projection is then

$$\alpha_t(d\theta) = \int_{|y-t| \leq l} \gamma H(d\theta) dy \beta(dl)/2l = \gamma H(d\theta) \int_0^\infty \beta(dl) \int_{t-l}^{t+l} dy/2l = \gamma H(d\theta) \tag{9}$$

so that each $D_t$ is again simply $DP(\gamma H)$. Now $D_s$ and $D_t$ will always be dependent with the amount of dependence decreasing as $|s - t|$ increases.

## 3.2 Interpretation as Mixtures of DPs

Even though the SN$\Gamma$P as described above defines an uncountably infinite number of DPs, in practice we will only have observations at a finite number of times, say $t_1, \ldots, t_m$. We define $\mathcal{R}$ as the smallest collection of disjoint *regions* of $\mathbb{Y}$ such that each $Y_{t_j}$ is a union of subsets in $\mathcal{R}$. Thus $\mathcal{R} = \{\cap_{j=1}^m S_j : S_j = Y_{t_j} \text{ or } S_j = \mathbb{Y} \backslash Y_{t_j}, \text{ with at least one } S_j = Y_{t_j} \text{ and } \cap_{j=1}^m S_j \neq \emptyset\}$. For $1 \leq j \leq m$ let $\mathcal{R}_j$ be the collection of regions in $\mathcal{R}$ contained in $Y_{t_j}$, so that $\cup_{R \in \mathcal{R}_j} = Y_{t_j}$. For each $R \in \mathcal{R}$ define

$$G_R(d\theta) = G(d\theta \otimes R) \tag{10}$$

We see that each $G_R$ is a $\Gamma$P with base measure $\alpha_R(d\theta) = \alpha(d\theta \otimes R)$. Normalizing, $D_R = G_R/G_R(\Theta) \sim DP(\alpha_R)$, with $D_R \perp\!\!\!\perp D_{R'}$ for distinct $R, R' \in \mathcal{R}$. Now

$$D_{t_j}(d\theta) = \sum_{R \in \mathcal{R}_j} \frac{G_R(\Theta)}{\sum_{R' \in \mathcal{R}_j} G_{R'}(\Theta)} D_R(d\theta) \tag{11}$$

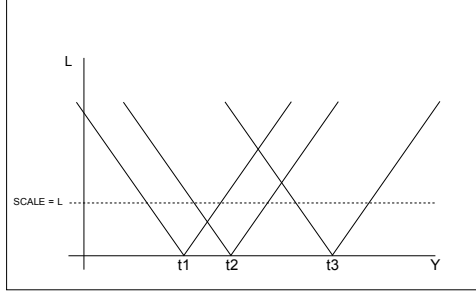

Figure 1: The extended space $\mathbb{Y} \otimes \mathbb{L}$ over which the overall $\Gamma$P is defined in the second example. Not shown is the $\Theta$-space over which the DPs are defined. Also not shown is the fourth dimension $W$ needed to define the Poisson process used to construct the $\Gamma$P. $t_1, t_2, t_3 \in \mathbb{Y}$ are three times at which observations are present. The subset $Y_{t_j}$ corresponding to each $t_j$ is the triangular area touching $t_j$. The regions in $\mathcal{R}$ are the six areas formed by various intersections of the triangular areas.

so each $D_{t_j}$ is a mixture where each component $D_R$ is drawn independently from a DP. Further, the mixing proportions are Dirichlet distributed and independent from the components by virtue of each $G_R(\Theta)$ being gamma distributed and independent from $D_R$. Thus we have the following equivalent construction for a SN$\Gamma$P:

$$D_R \sim \text{DP}(\alpha_R) \qquad g_R \sim \text{Gamma}(\alpha_R(\Theta)) \qquad \text{for } R \in \mathcal{R}$$

$$D_{t_j} = \sum_{R \in \mathcal{R}_j} \pi_{jR} D_R \qquad \pi_{jR} = \frac{g_R}{\sum_{R' \in \mathcal{R}_j} g_R} \qquad \text{for } R \in \mathcal{R}_j \qquad (12)$$

Note that the DPs in this construction are all defined only over $\Theta$, and references to the auxiliary space $\mathbb{Y}$ and the base measure $\alpha$ are only used to define the individual base measures $\alpha_R$ and the shape parameters of the $g_R$'s. Figure 1 shows the regions for the second example corresponding to observations at three times.

The mixture of DPs construction is related to the *hierarchical Dirichlet process* defined in [11] (not the one defined by Teh et al [3]). The difference is that the parameters of the prior over the mixing proportions exactly matches the concentration parameters of the individual DPs. A consequence of this is that each mixture $D_{t_j}$ is now conveniently also a DP.

# 4   Inference in the SN$\Gamma$P

The mixture of DPs interpretation of the SN$\Gamma$P makes sampling from the model, and consequently inference via Markov chain Monte Carlo sampling, easy. In what follows, we describe both Gibbs sampling and Metropolis-Hastings based updates for a hierarchical model in which the dependent DPs act as prior distributions over a collection of infinite mixture models. Formally, our observations now lie in a measurable space $(\mathbb{X}, \Sigma)$ equipped with a set of probability measures $F_\theta$ parametrized by $\theta \in \Theta$. Observation $i$ at time $t_j$ is denoted $x_{ji}$, lies in region $r_{ji}$ and is drawn from mixture component parametrized by $\theta_{ji}$. Thus to augment (12), we have

$$r_{ji} \sim \text{Mult}(\{\pi_{jR} : R \in \mathcal{R}_j\}) \qquad \theta_{ji} \sim D_{r_{ji}} \qquad x_{ji} \sim F_{\theta_{ji}} \qquad (13)$$

where $r_{ji} = R$ with probability $\pi_{jR}$ for each $R \in \mathcal{R}_j$. In words, we first pick a region $r_{ji}$ from the set $\mathcal{R}_j$, then a mixture component $\theta_{ji}$, followed by drawing $x_{ji}$ from the mixture distribution.

## 4.1   Gibb Sampling

We derive a Gibbs sampler for the SN$\Gamma$P where the region DPs $D_R$ are integrated out and replaced by Chinese restaurants. Let $c_{ji}$ denote the index of the cluster in $D_{r_{ji}}$ to which observation $x_{ji}$ is assigned. We also assume that the base distribution $H$ is conjugate to the mixture distributions $F_\theta$ so that the cluster parameters are integrated out as well. The Gibbs sampler iteratively resamples the

latent variables left: $r_{ji}$'s, $c_{ji}$'s and $g_R$'s. In the following, let $m_{jRc}$ be the number of observations from time $t_j$ assigned to cluster $c$ in the DP $D_R$ in region $R$, and let $f_{Rc}^{\neg ji}(x_{ji})$ be the density of observation $x_{ji}$ conditioned on the other variables currently assigned to cluster $c$ in $D_R$, with its cluster parameters integrated out. We denote marginal counts with dots, for example $m_{\cdot Rc}$ is the number of observations (over all times) assigned to cluster $c$ in region $R$. Superscripts $^{\neg ji}$ means observation $x_{ji}$ is excluded.

$r_{ji}$ and $c_{ji}$ are resampled together; their conditional joint probability given the other variables is:

$$p(r_{ji} = R, c_{ji} = c | \text{others}) \propto \left( \frac{g_R}{\sum_{r \in \mathcal{R}_j} g_r} \right) \left( \frac{m_{\cdot Rc}^{\neg ji}}{m_{\cdot R \cdot}^{\neg ji} + \alpha_R(\Theta)} \right) f_{Rc}^{\neg ji}(x_{ji}) \qquad (14)$$

The probability of $x_{ji}$ joining a new cluster in region $R$ is

$$p(r_{ji} = R, c_{ji} = c_{\text{new}} | \text{others}) \propto \left( \frac{g_R}{\sum_{r \in \mathcal{R}_j} g_r} \right) \left( \frac{\alpha_R(\Theta)}{m_{\cdot R \cdot}^{\neg ji} + \alpha_R(\Theta)} \right) f_{Rc_{\text{new}}}(x_{ji}) \qquad (15)$$

where $R \in \mathcal{R}_j$ and $c$ denotes the index of an existing cluster in region $R$. The updates of the $g_R$'s are more complicated as they are coupled and not of standard form:

$$p(\{g_R\}_{R \in \mathcal{R}} | \text{others}) = \left( \prod_{R \in \mathcal{R}} g_R^{\alpha_R(\Theta) + m_{\cdot R \cdot} - 1} e^{-g_R} \right) \prod_j \left( \sum_{R \in \mathcal{R}_j} g_R \right)^{-m_{j \cdot \cdot}} \qquad (16)$$

To sample the $g_R$'s we introduce auxiliary variables $\{A_j\}$ to simplify the rightmost term above. In particular, using the Gamma identity

$$\Gamma(m_{j \cdot \cdot}) \left( \sum_{R \in \mathcal{R}_j} g_R \right)^{-m_{j \cdot \cdot}} = \int_0^\infty A_j^{m_{j \cdot \cdot} - 1} e^{-\sum_{R \in \mathcal{R}_j} g_R A_j} dA_j \qquad (17)$$

we have that (16) is the marginal of $\{g_R\}_{R \in \mathcal{R}}$ of the distribution:

$$q(\{g_R\}_{R \in \mathcal{R}}, \{A_j\}) \propto \prod_{R \in \mathcal{R}} g_R^{\alpha_R(\Theta) + m_{\cdot R \cdot} - 1} e^{-g_R} \prod_j A_j^{m_{j \cdot \cdot} - 1} e^{-\sum_{R \in \mathcal{R}_j} g_R A_j} \qquad (18)$$

Now we can Gibbs sample the $g_R$'s and $A_j$'s:

$$g_R | \text{others} \sim \text{Gamma}(\alpha_R(\Theta) + m_{\cdot R \cdot}, 1 + \sum_{j \in J_R} A_j) \qquad (19)$$

$$A_j | \text{others} \sim \text{Gamma}(m_{j \cdot \cdot}, \sum_{R \in \mathcal{R}_j} g_R) \qquad (20)$$

Here $J_R$ is the collection of indices $j$ such that $R \in \mathcal{R}_j$.

## 4.2 Metropolis-Hastings Proposals

To improve convergence and mixing of the Markov chain, we introduce three Metropolis-Hastings (MH) proposals in addition to the Gibbs sampling updates described above. These propose non-incremental changes in the assignment of observations to clusters and regions, allowing the Markov chain to traverse to different modes that are hard to reach using Gibbs sampling.

The first proposal (Algorithm 1) proceeds like the split-merge proposal of [12]. It either splits an existing cluster in a region into two new clusters in the *same* region, or merges two existing clusters in a region into a single cluster. To improve the acceptance probability, we use 5 rounds of restricted Gibbs sampling [12].

The second proposal (Algorithm 2) seeks to move a picked cluster from one region to another. The new region is chosen from a region neighbouring the current one (for example in Figure 1 the neigbors are the four regions diagonally neighbouring the current one). To improve acceptance probability we also resample the $g_R$'s associated with the current and proposed regions. The move can be invalid if the cluster contains an observation from a time point not associated with the new region; in this case the move is simply rejected.

The third proposal (Algorithm 3) we considered seeks to combine into one step what would take two steps under the previous two proposals: splitting a cluster and moving it to a new region (or the reverse: moving a cluster into a new region and merging it with a cluster therein).

---
**Algorithm 1** Split and Merge in the Same Region (MH1)
---
1: Let $S_0$ be the current state of the Markov chain.
2: Pick a region $R$ with probability proportional to $m_{\cdot R\cdot}$ and two distinct observations in $R$
3: Construct a launch state $S'$ by creating two new clusters, each containing one of the two observations, and running restricted Gibbs sampling
4: **if** the two observations belong to the same cluster in $S_0$ **then**
5:     Propose split: run one last round of restricted Gibbs sampling to reach the proposed state $S_1$
6: **else**
7:     Propose merge: the proposed state $S_1$ is the (unique) state merging the two clusters
8: **end if**
9: Accept proposed state $S_1$ according to acceptance probability $\min\left(1, \frac{p(S_1)q(S' \to S_0)}{p(S_0)q(S' \to S_1)}\right)$ where

   $p(S)$ is the posterior probability of state $S$ and $q(S' \to S)$ is the probability of proposing state $S$ from the launch state $S'$.
---

---
**Algorithm 2** Move (MH2)
---
1: Pick a cluster $c$ in region $R_0$ with probability proportional to $m_{\cdot R_0 c}$
2: Pick a region $R_1$ neighbouring $R_0$ and propose moving $c$ to $R_1$
3: Propose new weights $g_{R_0}, g_{R_1}$ by sampling both from (19)
4: Accept or reject the move
---

---
**Algorithm 3** Split/merged Move (MH3)
---
1: Pick a region $R_0$, a cluster $c$ contained in $R$, and a neighbouring region $R_1$ with probability proportional to the number of observations in $c$ that *cannot* be assigned to a cluster in $R_1$
2: **if** $c$ contains observations than can be moved to $R_1$ **then**
3:     Propose assigning these observations to a new cluster in $R_1$
4: **else**
5:     Pick a cluster from those in $R_1$ and propose merging it into $c$
6: **end if**
7: Propose new weights $g_{R_0}, g_{R_1}$ by sampling from (19)
8: Accept or reject the proposal
---

## 5   Experiments

**Synthetic data**   In the first of our experiments, we artificially generated 60 data points at each of 5 times by sampling from a mixture of 10 Gaussians. Each component was assigned a timespan, ranging from a single time to the entire range of five times. We modelled this data as a collection of five DP mixture of Gaussians, with a SNΓP prior over the five dependent DPs. We used the set-up as described in the second example. To encourage clusters to be shared across times (i.e. to avoid similar clusters with non-overlapping timespans), we chose the distribution over window lengths $\beta(w)$ to give larger probabilities to larger timespans. Even in this simple model, Gibbs sampling alone usually did not converge to a good optimum; remaining stuck around local maxima. Figure 2 shows the evolution of the log-likelihood for 5 different samplers: plain Gibbs sampling, Gibbs sampling augmented with each of MH proposals 1, 2 and 3, and finally a sampler that interleaved all three MH samplers with Gibbs sampling. Not surprisingly, the complete sampler converged fastest, with Gibbs sampling with MH-proposal 2 (Gibbs+MH2) performing nearly as well. Gibbs+MH1 seemed converge no faster than just Gibbs sampling, with Gibbs+MH3 giving performance somewhere in between. The fact that Gibbs+MH2 performs so well can be explained by the easy clustering structure of the problem, so that exploring region assignments of clusters rather than cluster assignments of observations was the challenge faced by the sampler (note its high acceptance rate in Figure 4).

To demonstrate how the additional MH proposals help mixing, we examined how the cluster assignment of observations varied over iterations. At each iteration, we construct a 600 by 600 binary matrix, with element $(i, j)$ being 1 if observations $i$ and $j$ are assigned to the same cluster. In Figure 3, we plot the average $L_1$ difference between matrices at different iteration lags. Somewhat counterintuitively, Gibbs+MH1 does much better than Gibbs sampling with *all* MH proposals.

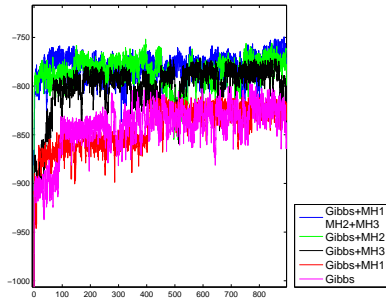

Figure 2: log-likelihoods (the coloured lines are ordered at iteration 80 like the legend).

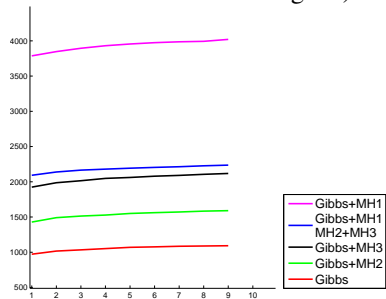

Figure 3: Dissimilarity in clustering structure vs lag (the coloured lines are ordered like the legend).

Figure 4: Acceptance rates of the MH proposals for Gibbs+MH1+MH2+MH3 after burn-in (percentages).

| Proposal | Synthetic | NIPS |
|---|---|---|
| MH-Proposal 1 | 0.51 | 0.6621 |
| MH-Proposal 2 | 11.7 | 0.6548 |
| MH-Proposal 3 | 0.22 | 0.0249 |

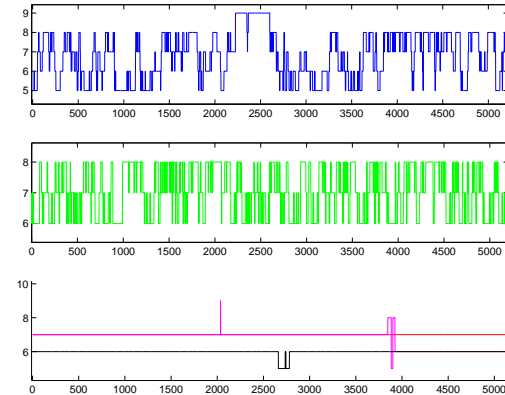

Figure 5: Evolution of the timespan of a cluster. From top to bottom: Gibbs+MH1+MH2+MH3, Gibbs+MH2 and Gibbs+MH1 (pink), Gibbs+MH3 (black) and Gibbs (magenta).

This is because the latter is simultaneously exploring the region assignment of clusters as well. In Gibbs+MH1, clusters split and merge frequently since they stay in the same regions, causing the cluster matrix to vary rapidly. In Gibbs+MH1+MH2+MH3, after a split the new clusters often move into separate regions; so it takes longer before they can merge again. Nonetheless, this demonstrates the importance of split/merge proposals like MH1 and MH3; [12] studied this in greater detail. We next examined how well the proposals explore the region assignment of clusters. In particular, at each step of the Markov chain, we pick the cluster with mean closest to the mean of one of the true Gaussian mixture components, and tracked how its timespan evolved. Figure 5 shows that without MH proposal 2, the clusters remain essentially frozen in their initial regions.

**NIPS dataset** For our next experiment we modelled the proceedings of the first 13 years of NIPS. The number of word tokens was about 2 million spread over 1740 documents, with about 13000 unique words. We used a model that involves both the SNΓP (to capture changes in topic distributions across the years) and the hierarchical Dirichlet process (HDP) [3] (to capture differences among documents). Each document is modeled using a different DP, with the DPs in year $i$ sharing the same base distribution $D_i$. On top of this, we place a SNΓP (with structure given by the second example in Section 3.1) prior on $\{D_i\}_{i=1}^{13}$. Consequently, each topic is associated with a distribution over words, and has a particular timespan. Each document in year $i$ is a mixture over the topics whose timespan include year $i$. Our model allows statistical strength to be shared in a more refined manner than the HDP. Instead of all DPs having the same base distribution, we have 13 dependent base distributions drawn from the SNΓP. The concentration parameters of our DPs were chosen to encourage shared topics, their magnitude chosen to produce about a 100 topics over the whole corpus on average. Figure 6 shows some of the topics identified by the model and their timespans. For inference, we used Gibbs sampling, interleaved with all three MH proposals to update the SNΓP. the Markov chain was initialized randomly except that all clusters were assigned to the top-most region (spanning the 13 years). We calculated per-word perplexity [3] on test documents (about half of all documents, withheld during training). We obtained an average perplexity of 3023.4, as opposed to about 3046.5 for the HDP.

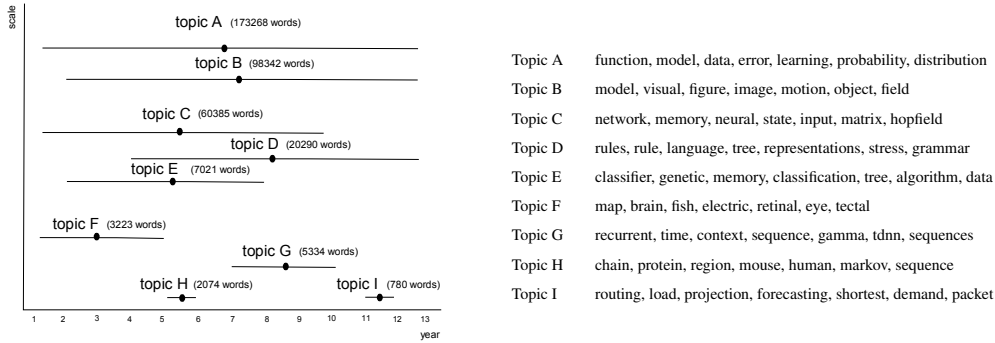

Figure 6: Inferred topics with their timespans (the horizontal lines). In parentheses are the number of words assigned to each topic. On the right are the top ten most probable words in the topics.

Computationally, the 3 MH steps are much cheaper than a round of Gibbs sampling. When trying to split a large cluster (or merge 2 large clusters), MH proposal 1 can still be fairly expensive because of the rounds of restricted Gibbs sampling. MH proposal 3 does not face this problem. However we find that after the burn-in period it tends to have low acceptance rate. We believe we need to redesign MH proposal 3 to produce more intelligent splits to increase the acceptance rate. Finally, MH-proposal 2 is the cheapest, both in terms of computation and book-keeping, and has reasonably high acceptance rate. We ran MH-proposal 2 a hundred times between successive Gibbs sampling updates. The acceptance rates of the MH proposals (given in Figure 4) are slightly lower than those reported by [12], where a plain DP mixture model was applied to a simple synthetic data set, and where split/merge acceptance rates were on the order of 1 to 5 percent.

## 6 Discussion

We described a conceptually simple and elegant framework for the construction of dependent DPs based on normalized gamma processes. The resulting collection of random probability measures has a number of useful properties: the marginal distributions are DPs and the weights of shared atoms can vary across DPs. We developed auxiliary variable Gibbs and Metropolis-Hastings samplers for the model and applied it to time-varying topic modelling where each topic has its own time-span.

Since [6] there has been strong interest in building dependent sets of random measures. Interestingly, the property of each random measure being marginally DP, as originally proposed by [6], is often not met in the literature, where dependent stochastic processes are defined through shared and random parameters [3, 14, 15, 11]. Useful dependent DPs had not been found [16] until recently, when a flurry of models were proposed [17, 18, 19, 20, 21, 22, 23]. However most of these proposals have been defined only for the real line (interpreted as the time line) and not for arbitrary spaces. [24, 25, 26, 13] proposed a variety of spatial DPs where the atoms and weights of the DPs are dependent through Gaussian processes. A model similar to ours was proposed recently in [23], using the same basic idea of introducing dependencies between DPs through spatially overlapping regions. This model differs from ours in the content of these shared regions (breaks of a stick in that case vs a (restricted) Gamma process in ours) and the construction of the DPs (they use the stick breaking construction of the DP, we normalize the restricted Gamma process). Consequently, the nature of the dependencies between the DPs differ; for instance, their model cannot be interpreted as a mixture of DPs like ours.

There are a number of interesting future directions. First, we can allow, at additional complexity, the locations of atoms to vary using the spatial DP approach [13]. Second, more work need still be done to improve inference in the model, e.g. using a more intelligent MH proposal 3. Third, although we have only described spatial normalized gamma processes, it should be straightforward to extend the approach to spatial normalized random measures [7, 8]. Finally, further investigations into the properties of the SNΓP and its generalizations, including the nature of the dependency between DPs and asymptotic behavior, are necessary for a complete understanding of these processes.

# References

[1] T. S. Ferguson. A Bayesian analysis of some nonparametric problems. *Annals of Statistics*, 1(2):209–230, 1973.

[2] M. J. Beal, Z. Ghahramani, and C. E. Rasmussen. The infinite hidden Markov model. In *Advances in Neural Information Processing Systems*, volume 14, 2002.

[3] Y. W. Teh, M. I. Jordan, M. J. Beal, and D. M. Blei. Hierarchical Dirichlet processes. *Journal of the American Statistical Association*, 101(476):1566–1581, 2006.

[4] C. Kemp, J. B. Tenenbaum, T. L. Griffiths, T. Yamada, and N. Ueda. Learning systems of concepts with an infinite relational model. In *Proceedings of the AAAI Conference on Artificial Intelligence*, volume 21, 2006.

[5] M. Johnson, T. L. Griffiths, and S. Goldwater. Adaptor grammars: A framework for specifying compositional nonparametric Bayesian models. In *Advances in Neural Information Processing Systems*, volume 19, 2007.

[6] S. MacEachern. Dependent nonparametric processes. In *Proceedings of the Section on Bayesian Statistical Science*. American Statistical Association, 1999.

[7] L. E. Nieto-Barajas, I. Pruenster, and S. G. Walker. Normalized random measures driven by increasing additive processes. *Annals of Statistics*, 32(6):2343–2360, 2004.

[8] L. F. James, A. Lijoi, and I. Pruenster. Bayesian inference via classes of normalized random measures. ICER Working Papers - Applied Mathematics Series 5-2005, ICER - International Centre for Economic Research, April 2005.

[9] J. F. C. Kingman. Completely random measures. *Pacific Journal of Mathematics*, 21(1):59–78, 1967.

[10] J. F. C. Kingman. *Poisson Processes*. Oxford University Press, 1993.

[11] P. Müller, F. A. Quintana, and G. Rosner. A method for combining inference across related nonparametric Bayesian models. *Journal of the Royal Statistical Society*, 66:735–749, 2004.

[12] S. Jain and R. M. Neal. A split-merge Markov chain Monte Carlo procedure for the Dirichlet process mixture model. Technical report, Department of Statistics, University of Toronto, 2004.

[13] J. A. Duan, M. Guindani, and A. E. Gelfand. Generalized spatial Dirichlet process models. *Biometrika*, 94(4):809–825, 2007.

[14] A. Rodríguez, D. B. Dunson, and A. E. Gelfand. The nested Dirichlet process. Technical Report 2006-19, Institute of Statistics and Decision Sciences, Duke University, 2006.

[15] D. B. Dunson, Y. Xue, and L. Carin. The matrix stick-breaking process: Flexible Bayes meta analysis. Technical Report 07-03, Institute of Statistics and Decision Sciences, Duke University, 2007. http://ftp.isds.duke.edu/WorkingPapers/07-03.html.

[16] N. Srebro and S. Roweis. Time-varying topic models using dependent Dirichlet processes. Technical Report UTML-TR-2005-003, Department of Computer Science, University of Toronto, 2005.

[17] J. E. Griffin and M. F. J. Steel. Order-based dependent Dirichlet processes. *Journal of the American Statistical Association, Theory and Methods*, 101:179–194, 2006.

[18] J. E. Griffin. The Ornstein-Uhlenbeck Dirichlet process and other time-varying processes for Bayesian nonparametric inference. Technical report, Department of Statistics, University of Warwick, 2007.

[19] F. Caron, M. Davy, and A. Doucet. Generalized Polya urn for time-varying Dirichlet process mixtures. In *Proceedings of the Conference on Uncertainty in Artificial Intelligence*, volume 23, 2007.

[20] A. Ahmed and E. P. Xing. Dynamic non-parametric mixture models and the recurrent Chinese restaurant process. In *Proceedings of The Eighth SIAM International Conference on Data Mining*, 2008.

[21] J. E. Griffin and M. F. J. Steel. Bayesian nonparametric modelling with the Dirichlet process regression smoother. Technical report, University of Kent and University of Warwick, 2008.

[22] J. E. Griffin and M. F. J. Steel. Generalized spatial Dirichlet process models. Technical report, University of Kent and University of Warwick, 2009.

[23] Y. Chung and D. B. Dunson. The local Dirichlet process. *Annals of the Institute of Mathematical Statistics*, 2009. to appear.

[24] S.N. MacEachern, A. Kottas, and A.E. Gelfand. Spatial nonparametric Bayesian models. In *Proceedings of the 2001 Joint Statistical Meetings*, 2001.

[25] C. E. Rasmussen and Z. Ghahramani. Infinite mixtures of Gaussian process experts. In *Advances in Neural Information Processing Systems*, volume 14, 2002.

[26] A. E. Gelfand, A. Kottas, and S. N. MacEachern. Bayesian nonparametric spatial modeling with Dirichlet process mixing. *Journal of the American Statistical Association*, 100(471):1021–1035, 2005.

